# The Ni1000: High Speed Parallel VLSI for Implementing Multilayer Perceptrons

**Michael P. Perrone**
Thomas J. Watson Research Center
P.O. Box 704
Yorktown Heights, NY 10598
mpp@watson.ibm.com

**Leon N Cooper**
Institute for Brain and Neural Systems
Brown University
Providence, Ri 02912
lnc@cns.brown.edu

## Abstract

In this paper we present a new version of the standard multilayer perceptron (MLP) algorithm for the state-of-the-art in neural network VLSI implementations: the Intel Ni1000. This new version of the MLP uses a fundamental property of high dimensional spaces which allows the $l_2$-norm to be accurately approximated by the $l_1$-norm. This approach enables the standard MLP to utilize the parallel architecture of the Ni1000 to achieve on the order of 40000, 256-dimensional classifications per second.

## 1 The Intel Ni1000 VLSI Chip

The Nestor/Intel radial basis function neural chip (Ni1000) contains the equivalent of 1024 256-dimensional artificial digital neurons and can perform at least 40000 classifications per second [Sullivan, 1993]. To attain this great speed, the Ni1000 was designed to calculate "city block" distances (i.e. the $l_1$-norm) and thus to avoid the large number of multiplication units that would be required to calculate Euclidean dot products in parallel. Each neuron calculates the city block distance between its stored weights and the current input:

$$\text{neuron activity} = \sum_i |w_i - x_i| \tag{1}$$

where $w_i$ is the neuron's stored weight for the $i$th input and $x_i$ is the $i$th input. Thus the Ni1000 is ideally suited to perform both the RCE [Reilly et al., 1982] and

PRCE [Scofield et al., 1987] algorithms or any of the other commonly used radial basis function (RBF) algorithms. However, dot products are central in the calculations performed by most neural network algorithms (e.g. MLP, Cascade Correlation, etc.). Furthermore, for high dimensional data, the dot product becomes the computation bottleneck (i.e. most of the network's time is spent calculating dot products). If the dot product can not be performed in parallel there will be little advantage using the Ni1000 for such algorithms. In this paper, we address this problem by showing that we can extend the Ni1000 to many of the standard neural network algorithms by representing the Euclidean dot product as a function of Euclidean norms and by then using a city block norm approximation to the Euclidean norm. Section 2, introduces the approximate dot product; Section 3 describes the City Block MLP which uses the approximate dot product; and Section 4 presents experiments which demonstrate that the City Block MLP performs well on the NIST OCR data and on human face recognition data.

## 2 Approximate Dot Product

Consider the following approximation [Perrone, 1993]:

$$||\vec{z}|| \approx \frac{1}{\sqrt{n}} |\vec{z}| \tag{2}$$

where $\vec{z}$ is some $n$-dimensional vector, $||\cdot||$ is the Euclidean length (i.e. the $l_2$-norm) and $|\cdot|$ is the City Block length (i.e. the $l_1$-norm). This approximation is motivated by the fact that in high dimensional spaces it is accurate for a majority of the points in the space. In Figure 1, we suggest an intuitive interpretation of why this approximation is reasonable. It is clear from Figure 1 that the approximation is reasonable for about 20% of the points on the arc in 2 dimensions.[1] As the dimensionality of the data space increases, the tangent region in Figure 1 expands asymptotically to fill the entire space and thus the approximation becomes more valid. Below we examine how accurate this approximation is and how we can use it with the Ni1000, particularly in the MLP context. Given a set of vectors, $V$, all with equal city block length, we measure the accuracy of the approximation by the ratio of the variance of the Euclidean lengths in $V$ to the squared mean Euclidean lengths in $V$. If the ratio is low, then the approximation is good and all we must do is scale the city block length to the mean Euclidean length to get a good fit.[2] In particular, it can be shown that assuming all the vectors of the space are equally likely, the following equation holds [Perrone, 1993]:

$$\sigma_n^2 < \left( \frac{2n}{\alpha_n^2(n+1)} - 1 \right) \mu_{\text{lower}}^2, \tag{3}$$

where $n$ is the dimension of the space; $\mu_n$ is the average Euclidean length of the set of vectors with fixed city block length $S$; $\sigma_n^2$ is the variance about the average Euclidean length; $\mu_{\text{lower}}$ is the lower bound for $\mu_n$ and is given by $\mu_{\text{lower}} \equiv \alpha_n S/\sqrt{n}$;

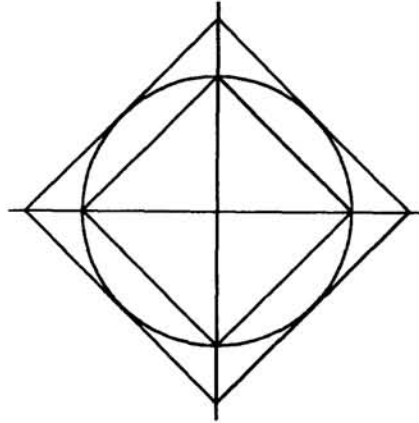

Figure 1: Two dimensional interpretation of the city block approximation. The circle corresponds to all of the vectors with the same Euclidean length. The inner square corresponds to all of the vectors with city block length equal the Euclidean length of the vectors in the circle. The outer square (tangent to the circle) corresponds to the set of vectors over which we will be making our approximation. In order to scale the outer square to the inner square, we multiple by $1/\sqrt{n}$ where $n$ is the dimensionality of the space. The outer square approximates the circle in the regions near the tangent points. In high dimensional spaces, these tangent regions approximate a large portion of the total hypersphere and thus the city block distance is a good approximation along most of the hypersphere.

and $\alpha_n$ is defined by

$$\alpha_n \equiv \frac{n-1}{n+1}\sqrt{1+\frac{en}{2\pi(n-1)}}\left(\frac{n}{2}\right)^{\frac{1}{2n-2}} + \frac{2}{n+1}. \tag{4}$$

From this equation we see that the ratio of $\sigma_n^2$ to $\mu_{\text{lower}}^2$ in the large $n$ limit is bounded above by 0.4. This bound is not very tight due to the complexity of the calculations required; however Figure 3 suggests that a much tighter bound must exist. A better bound exists if we are willing to add a minor constraint to our high dimensional space [Perrone, 1993]. In the case in which each dimension of the vector is constrained such that the entire vector cannot lie along a single axis,[3] we can show that

$$\sigma_n^2 \approx \frac{2(n-1)}{(n+1)^2}\left(\sqrt{\frac{n}{S}}-1\right)^2\frac{\mu_{\text{lower}}^2}{\alpha_n^2}, \tag{5}$$

where $S$ is the city block length of the vector in question. Thus in this case, the ratio of $\sigma_n^2$ to $\mu_{\text{lower}}^2$ decreases at least as fast as $1/n$ since $n/S$ will be some fixed constant independent of n.[4] This dependency on $n$ and $S$ is shown in Figure 2. This result suggests that the approximation will be very accurate for many real-world pattern

recognition tasks such as speech and high resolution image recognition which can typically have thousand or even tens of thousands of dimensions.

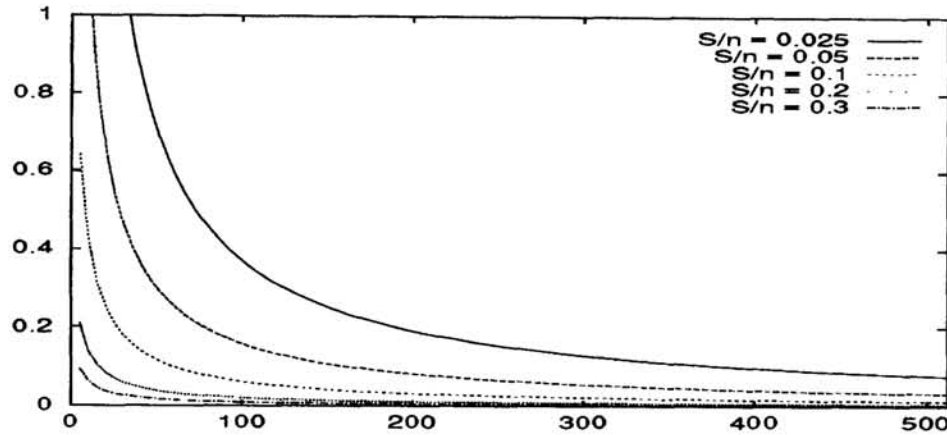

Figure 2: Plot of $\sigma_n/\mu_{\text{lower}}$ vs. $n$ for constrained vectors with varying values of $S/n$. As $S$ grows the ratio shrinks and consequently, accuracy improves. If we assume that all of the vectors are uniformly distributed in an $n$-dimensional unit hypercube, it is easy to show that the average city block length is $n/2$ and the variance of the city block length is $n/12$. Since $S/n$ will generally be within one standard deviation of the mean, we find that typically $0.2 < S/n < 0.8$. We can use the same analysis on binary valued vectors to derive similar results.

We explore this phenomenon further by considering the following Monte Carlo simulation. We sampled 200000 points from a uniform distribution over an $n$-dimensional cube. The Euclidean distance of each of these points to a fixed corner of the cube was calculated and all the lengths were normalized by the largest possible length, $\sqrt{n}$. Histograms of the resulting lengths are shown in Figure 3 for four different values of $n$. Note that as the dimension increases the variance about the mean drops. From Figure 3 we see that for as few as 100 dimensions, the standard deviation is approximately 5% of the mean length.

## 3   The City Block MLP

In this section we describe how the approximation explained in Section 2 can be used by the Ni1000 to implement MLPs in parallel. Consider the following formula for the dot product

$$\vec{x} \cdot \vec{y} = \frac{1}{4}(\|\vec{x} + \vec{y}\|^2 - \|\vec{x} - \vec{y}\|^2) \tag{6}$$

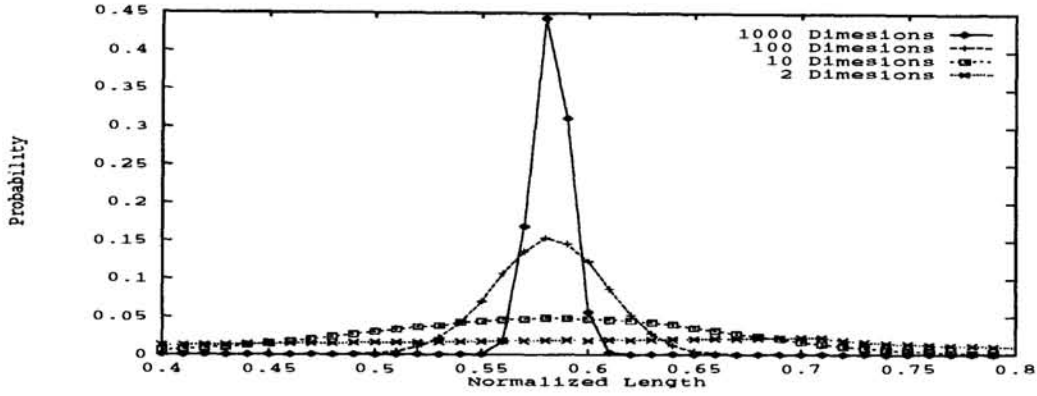

Figure 3: Probability distributions for randomly draw lengths. Note that as the dimension increases the variance about the mean length drops.

where $|| \cdot ||$ is the Euclidean length (i.e. $l_2$-norm).[5] Using Equation 2, we can approximation Equation 6 by

$$\vec{x} \cdot \vec{y} \approx \frac{1}{4n}(|\vec{x}+\vec{y}|^2 - |\vec{x}-\vec{y}|^2) \tag{7}$$

where $n$ is the dimension of the vectors and $| \cdot |$ is the city block length. The advantage to the approximation in Equation 7 is that it can be implemented in parallel on the Ni1000 while still behaving like a dot product. Thus we can use this approximation to implement MLP's on an Ni1000. The standard functional form for MLP's is given by [Rumelhart et al., 1986]

$$f_k(x;\alpha,\beta) = \sigma\left(\alpha_{0k} + \sum_{j=1}^{N} \alpha_{jk}\sigma(\beta_{0j} + \sum_{i=1}^{d} \beta_{ij}x_i)\right) \tag{8}$$

were $\sigma$ is a fixed ridge function chosen to be $\sigma(x) = (1 + e^{-x})^{-1}$; $N$ is the number of hidden units; $k$ is the class label; $d$ is the dimensionality of the data space; and $\alpha$ and $\beta$ are adjustable parameters. The alternative which we propose, the City Block MLP, is given by [Perrone, 1993]

$$g_k(x;\alpha,\beta) = \sigma\left(\alpha_{0k} + \sum_{j=1}^{N} \alpha_{jk}\sigma(\beta_{0j} + \frac{1}{4n}(\sum_{i=1}^{d} |\beta_{ij}+x_i|)^2 - \frac{1}{4n}(\sum_{i=1}^{d} |\beta_{ij}-x_i|)^2)\right) \tag{9}$$

$$\vec{x} \cdot \vec{y} = \frac{1}{2}(||\vec{x}+\vec{y}||^2 - ||\vec{x}||^2 - ||\vec{y}||^2)$$

or

$$\vec{x} \cdot \vec{y} = \frac{1}{2}(||\vec{x}||^2 + ||\vec{y}||^2 - ||\vec{x}-\vec{y}||^2).$$

| DATA SET | HIDDEN UNITS | STANDARD % CORRECT | CITYBLOCK % CORRECT | ENSEMBLE CITYBLOCK |
|---|---|---|---|---|
| Faces | 12 | 94.6±1.4 | 92.2±1.9 | 96.3 |
| Numbers | 10 | 98.4±0.17 | 97.3±0.26 | 98.3 |
| Lowercase | 20 | 88.9±0.31 | 84.0±0.48 | 88.6 |
| Uppercase | 20 | 90.5±0.39 | 85.6±0.89 | 90.7 |

Table 1: Comparison of MLPs classification performance with and with out the city block approximation to the dot product. The final column shows the effect of function space averaging.

where the two city block calculation would be performed by neurons on the Ni1000 chip.[6] The City Block MLP learns in the standard way by minimizing the mean square error (MSE),

$$\text{MSE} = \sum_{ik} \big(g_k(x_i; \alpha, \beta) - t_{ki}\big)^2 \tag{10}$$

where $t_{ki}$ is the value of the data at $x_i$ for a class $k$. The MSE is minimized using the backpropagation stochastic gradient descent learning rule [Werbos, 1974]: For a fixed stepsize $\eta$ and each $k$, randomly choose a data point $x_i$ and change $\gamma$ by the amount

$$\Delta\gamma = -\eta \frac{\partial(\text{MSE}_i)}{\partial\gamma}, \tag{11}$$

where $\gamma$ is either $\alpha$ or $\beta$ and $\text{MSE}_i$ is the contribution to the MSE of the $i$th data point. Note that although we have motivated the City Block MLP above as an approximation to the standard MLP, the City Block MLP can also be thought of as special case of radial basis function network.

## 4 Experimental Results

This section describes experiments using the City Block MLP on a 120-dimensional representation of the NIST Handwritten Optical Character Recognition database and on a 2294-dimensional grayscale human face image database. The results indicate that the performance of networks using the approximation is as good as networks using the exact dot product [Perrone, 1993].

In order to test the performance of the City Block MLP, we simulated the behavior of the Ni1000 on a SPARC station in serial. We used the approximation only on the first layer of weights (i.e. those connecting the inputs to the hidden units) where the dimensionality is highest and the approximation is most accurate. The approximation was not used in the second layer of weights (i.e. those connecting the hidden units to the output units were calculated in serial) since the number of hidden units was low and therefore do not correspond to a major computational bottleneck. It should be noted that for a 2 layer MLP in which the number of hidden units and output units are much lower than the input dimensionality, the

| DATA SET | HIDDEN UNITS | STANDARD FOM | CITYBLOCK FOM | ENSEMBLE CITYBLOCK |
|---|---|---|---|---|
| Numbers | 10 | 92.1±0.57 | 87.4±0.83 | 92.5 |
| Lowercase | 20 | 59.7±1.7 | 44.4±2.0 | 62.7 |
| Uppercase | 20 | 60.0±1.8 | 44.6±4.5 | 66.4 |

Table 2: Comparison of MLPs FOM. The FOM is defined as the 100 minus the number rejected minus 10 time the number incorrect.

majority of the computation is in the calculation of the dot products in the first weight layer. So even using the approximation only in the first layer will significantly accelerate the calculation. Also, the Ni1000 on-chip math coprocessor can perform a low-dimensional, second layer dot product while the high-dimensional, first layer dot product is being approximated in parallel by the city block units. In practice, if the number of hidden units is large, the approximation to the dot product may also be used in the second weight layer. In the simulations, the networks used the approximation when calculating the dot product only in the feedforward phase of the algorithm. For the feedbackward phase (i.e. the error backpropagation phase), the algorithm was identical to the original backward propagation algorithm. In other words the approximation was used to calculate the network activity but the stochastic gradient term was calculated as if the network activity was generated with the real dot product. This simplification does not slow the calculation because all the terms needed for the backpropagation phase are calculated in the forward propagation phase In addition, it allows us to avoid altering the backpropagation algorithm to incorporate the derivative of the city block approximation. We are currently working on simulations which use city block calculations in both the forward and backward passes. Since these simulations will use the correct derivative for the functional form of the City Block MLP, we expect that they will have better performance. In practice, the price we pay for making the approximation is reduced performance. We can avoid this problem by increasing the number of hidden units and thereby allow more flexibility in the network. This increase in size will not significantly slow the algorithm since the hidden unit activities are calculated in parallel. In Table 1 and Table 2, we compare the performance of a standard MLP without the city block approximation to a MLP using the city block approximation to calculate network activity. In all cases, a population of 10 neural networks were trained from random initial weight configurations and the means and standard deviations were listed. The number of hidden units was chosen to give a reasonable size network while at the same time reasonably quick training. Training was halted by cross-validating on an independent hold-out set. From these results, one can see that the relative performances with and with out the approximation are similar although the City Block is slightly lower. We also perform ensemble averaging [Perrone, 1993, Perrone and Cooper, 1993] to further improve the performance of the approximate networks. These results are given in the last column of the table. From these data we see that by combining the city block approximation with the averaging method, we can generate networks which have comparable and sometimes better performance than the standard MLPs. In addition, because the Ni1000 is running in parallel, there is minimal additional computational overhead for using

the averaging.[7]

## 5   Discussion

We have described a new radial basis function network architecture which can be used in high dimensional spaces to approximate the learning characteristics of a standard MLP without using dot products. The absence of dot products allows us to implement this new architecture efficiently in parallel on an Ni1000; thus enabling us to take advantage of the Ni1000's extremely fast classification rates. We have also presented experimental results on real-world data which indicate that these high classifications rates can be achieved while maintaining or improving classification accuracy. These results illustrate that it is possible to use the inherent high dimensionality of real-world problems to our advantage.

## Footnotes

[1]In fact, approximately 20% of the points are within 1% of each other and 40% of the points are within 5% of each other.

[2]Note that in Equation 2 we scale by $1/\sqrt{n}$. For high dimensional spaces this is a good approximation to the ratio of the mean Euclidean length to the City Block length.

[3] For example, when the axes are constrained to be in the range $[0,1]$ and the city block length of the vector is greater than 1. Note that this is true for the majority of the points in a $n$ dimensional unit hypercube.

[4] Thus the accuracy improves as $S$ increases towards its maximum value.

[5]Note also that depending on the information available to us, we could use either

[6]The dot product between the hidden and the output layers may also be approximated in the same way but it is not shown here. In fact, the Ni1000 could be used to perform all of the functions required by the network.

[7]The averaging can also be applied to the standard MLPs with a corresponding improvement in performance. However, for serial machines averaging slows calculations by a factor equal to the number of averaging nets.

## References

[Perrone, 1993] Perrone, M. P. (1993). *Improving Regression Estimation: Averaging Methods for Variance Reduction with Extensions to General Convex Measure Optimization.* PhD thesis, Brown University, Institute for Brain and Neural Systems; Dr. Leon N Cooper, Thesis Supervisor.

[Perrone and Cooper, 1993] Perrone, M. P. and Cooper, L. N. (1993). When networks disagree: Ensemble method for neural networks. In Mammone, R. J., editor, *Artificial Neural Networks for Speech and Vision.* Chapman-Hall. Chapter 10.

[Reilly et al., 1982] Reilly, D. L., Cooper, L. N., and Elbaum, C. (1982). A neural model for category learning. *Biological Cybernetics*, 45:35–41.

[Rumelhart et al., 1986] Rumelhart, D. E., McClelland, J. L., and the PDP Research Group (1986). *Parallel Distributed Processing, Volume 1: Foundations.* MIT Press.

[Scofield et al., 1987] Scofield, C. L., Reilly, D. L., Elbaum, C., and Cooper, L. N. (1987). Pattern class degeneracy in an unrestricted storage density memory. In Anderson, D. Z., editor, *Neural Information Processing Systems.* American Institute of Physics.

[Sullivan, 1993] Sullivan, M. (1993). Intel and Nestor deliver second-generation neural network chip to DARPA: Companies launch beta site program. *Intel Corporation News Release.* Feb. 12.

[Werbos, 1974] Werbos, P. (1974). *Beyond Regression: New Tools for Prediction and Analysis in the Behavioral Sciences.* PhD thesis, Harvard University.

